# Towards Faster Stochastic Gradient Search

**Christian Darken and John Moody**
Yale Computer Science, P.O. Box 2158, New Haven, CT 06520
Email: darken@cs.yale.edu

## Abstract

Stochastic gradient descent is a general algorithm which includes LMS, on-line backpropagation, and adaptive k-means clustering as special cases. The standard choices of the learning rate $\eta$ (both adaptive and fixed functions of time) often perform quite poorly. In contrast, our recently proposed class of "search then converge" learning rate schedules (Darken and Moody, 1990) display the *theoretically optimal* asymptotic convergence rate and a superior ability to escape from poor local minima. However, the user is responsible for setting a key parameter. We propose here a new methodology for creating the first completely automatic adaptive learning rates which achieve the *optimal rate of convergence*.

## Introduction

The stochastic gradient descent algorithm is

$$\Delta W(t) = -\eta \nabla_W E(W(t), X(t)).$$

where $\eta$ is the learning rate, $t$ is the "time", and $X(t)$ is the independent random exemplar chosen at time $t$. The purpose of the algorithm is to find a parameter vector $W$ which minimizes a function $G(W)$ which for learning algorithms has the form $\mathcal{E}_X E(W, X)$, i.e. $G$ is the average of an objective function over the exemplars, labeled $E$ and $X$ respectively. We can rewrite $\Delta W(t)$ in terms of $G$ as

$$\Delta W(t) = -\eta[\nabla_W G(W(t)) + \xi(t, W(t))],$$

where the $\xi$ are independent zero-mean noises. Stochastic gradient descent may be preferable to deterministic gradient descent when the exemplar set is increasing in size over time or large, making the average over exemplars expensive to compute.

Additionally, the noise in the gradient can help the system escape from local minima. The fundamental algorithmic issue is **how to best adjust $\eta$ as a function of time and the exemplars?**

## State of the Art Schedules

The usual non-adaptive choices of $\eta$ (i.e. $\eta$ depends on the time only) often yield poor performance. The simple expedient of taking $\eta$ to be constant results in persistent residual fluctuations whose magnitude and the resulting degradation of system performance are difficult to anticipate (see fig. 3). Taking a smaller constant $\eta$ reduces the magnitude of the fluctuations, but seriously slows convergence and causes problems with metastable local minima. Taking $\eta(t) = c/t$, the common choice in the stochastic approximation literature of the last forty years, typically results in slow convergence to bad solutions for small $c$, and parameter blow-up for small $t$ if $c$ is large (Darken and Moody, 1990).

The available adaptive schedules (i.e. $\eta$ depends on the time *and* on previous exemplars) have problems as well. Classical methods which involve estimating the hessian of G are often unusable because they require $O(N^2)$ storage and computation for each update, which is too expensive for large $N$ (many parameter systems— e.g. large neural nets). Methods such as those of Fabian (1960) and Kesten (1958) require the user to specify an entire function and thus are not practical methods as they stand. The delta-bar-delta learning rule, which was developed in the context of deterministic gradient descent (Jacobs, 1988), is often useful in locating the general vicinity of a solution in the stochastic setting. However it hovers about the solution without converging (see fig. 4). A schedule developed by Urasiev is proven to converge in principle, but in practice it converges slowly if at all (see fig. 5). The literature is widely scattered over time and disciplines, however to our knowledge no published $O(N)$ technique attains the optimal convergence speed.

## Search-Then-Converge Schedules

Our recently proposed solution is the "search then converge" learning rate schedule. $\eta$ is chosen to be a fixed function of time such as the following:

$$\eta(t) = \eta_0 \frac{1 + \frac{c}{\eta_0} \frac{t}{\tau}}{1 + \frac{c}{\eta_0} \frac{t}{\tau} + \tau \frac{t^2}{\tau^2}}$$

This function is approximately constant with value $\eta_0$ at times small compared to $\tau$ (the "search phase"). At times large compared with $\tau$ (the "converge phase"), the function decreases as $c/t$. See for example the eta vs. time curves for figs. 6 and 7. This schedule has demonstrated a dramatic improvement in convergence speed and quality of solution as compared to the traditional fixed learning rate schedule for k-means clustering (Darken and Moody, 1990). However, these benefits apply to supervised learning as well. Compare the error curve of fig. 3 with those of figs. 6 and 7.

This schedule yields optimally fast asymptotic convergence if $c > c^*$, $c^* \equiv 1/2\alpha$, where $\alpha$ is the smallest eigenvalue of the hessian of the function G (defined above)

**Little Drift**                          **Much Drift**

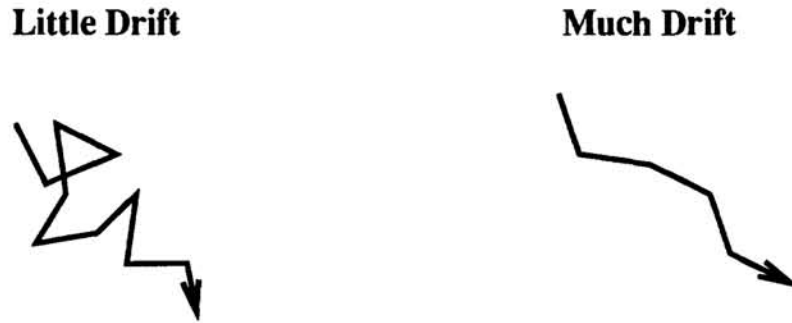

Figure 1: Two contrasting parameter vector trajectories illustrating the notion of drift

at the pertinent minimum (Fabian, 1968) (Major and Revesz, 1973) (Goldstein, 1987). The penalty for choosing $c < c^*$ is that the ratio of the excess error given $c$ too small to the excess error with c large enough gets arbitrarily large as training time grows, i.e.

$$\lim_{t \to \infty} \frac{E_{c<c^*}}{E_{c>c^*}} = \infty,$$

where $E$ is the excess error above that at the minimum. The same holds for the ratio of the two distances to the location of the minimum in parameter space.

While the above schedule works well, its asymptotic performance depends upon the user's choice of $c$. Since neither $\eta_0$ nor $\tau$ affects the asymptotic behavior of the system, we will discuss their selection elsewhere. Setting $c > c^*$, however, is vital. Can such a $c$ be determined automatically? Directly estimating $\alpha$ with conventional methods (by calculating the smallest eigenvalue of the hessian at our current estimate of the minimum) is too computationally demanding. This would take at least $O(N^2)$ storage and computation time for each estimate, and would have to be done repeatedly ($N$ is the number of parameters). We are investigating the possibility of a low complexity direct estimation of $\alpha$ by performing a second optimization. However here we take a more unusual approach: we shall determine whether $c$ is large enough by observing the trajectory of the parameter (or "weight") vector.

## On-line Determination of Whether $c < c^*$

We propose that excessive correlation in the parameter change vectors (i.e. "drift") indicates that $c$ is too small (see fig. 1). We define the drift as

$$D(t) \equiv \sum_k d_k^2(t)$$

$$d_k(t) \equiv \sqrt{T} \frac{\langle \delta_k(t) \rangle_T}{[\langle (\delta_k(t) - \langle \delta_k(t) \rangle_T)^2 \rangle_T]^{1/2}}$$

where $\delta_k(t)$ is the change in the $k$th component of the parameter vector at time $t$ and the angled brackets denote an average over $T$ parameter changes. We take $T = at$, where $a \ll 1$. Notice that the numerator is the average parameter step while the

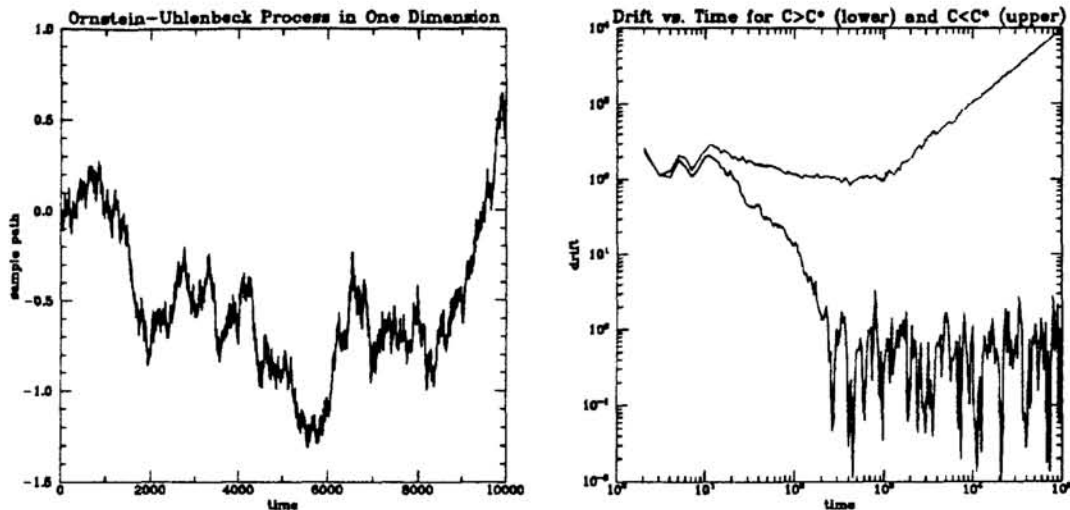

Figure 2: (Left) An Ornstein–Uhlenbeck process. This process is zero-mean, gaussian, and stationary (in fact strongly ergodic). It may be thought of as a random walk with a restoring force towards zero. (Right) Measurement of the drift for the runs $c = .1c^*$ and $c = 10c^*$ which are discussed in figs. 7 and 8 below.

denominator is the standard deviation of the steps. As a point of reference, if the $\delta_k$ were independent normal random variables, then the $d_k$ would be "T-distributed" with $T$ degrees of freedom, i.e. approximately unit-variance normals for moderate to large $T$. We find that $\delta_k$ may also be taken to be the $k$th component of the noisy gradient to the same effect.

Asymptotically, we will take the learning rate to go as $c/t$. Choosing $c$ too small results in a slow drift of the parameter vector towards the solution in a relatively linear trajectory. When $c > c^*$ however, the trajectory is much more jagged. Compare figs. 7 and 8. More precisely, we find that $D(t)$ **blows up like a power of $t$ when $c$ is too small, but remains finite otherwise.** Our experiments confirm this (for an example, see fig. 2). This provides us with a signal to use in future adaptive learning rate schemes for ensuring that $c$ is large enough.

The bold-printed statement above implies that an arbitrarily small change in $c$ which moves it to the opposite side of $c^*$ has dramatic consequences for the behavior of the drift. The following rough argument outlines how one might prove this statement, focusing on the source of this interesting discontinuity in behavior. We simplify the argument by taking the $\delta_k$'s to be gradient measurements as mentioned above. We consider a one-dimensional problem, and modify $d_1$ to be $\sqrt{T}\langle\delta_1\rangle_T$ (i.e. we ignore the denominator). Then since $T = at$ as stated above, we approximate

$$d_1 \equiv \sqrt{T}\langle\delta_1(t)\rangle_T \approx \langle\sqrt{t}\delta_1(t)\rangle_T = \langle\sqrt{t}[\nabla G(t) + \xi(t)]\rangle_T$$

Recall the definitions of $G$ and $\xi$ from the introduction above. As $t \to \infty$, $\nabla G(t) \to K[W(t) - W_0]$ for the appropriate $K$ by the Taylor's expansion for $G$ around $W_0$, the location of the local minimum. Thus

$$\lim_{t\to\infty} d_1 \approx \langle K\sqrt{t}[W(t) - W_0]\rangle_T + \langle\sqrt{t}\xi(t)\rangle_T$$

Define $X(t) \equiv \sqrt{t}[W(t) - W_0]$. Now according to (Kushner, 1978), $X(e^t)$ converges

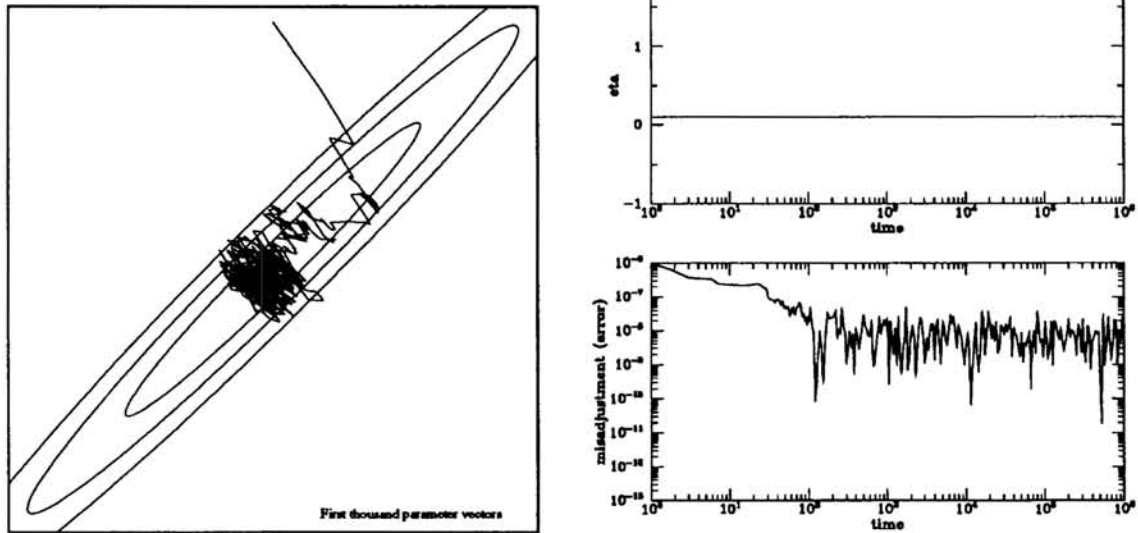

**Constant η=0.1**

Figure 3: The constant $\eta$ schedule, commonly used in training backpropagation networks, does not converge in the stochastic setting.

in distribution to the well-known Ornstein-Uhlenbeck process (fig. 2) when $c > c^*$. By extending this work, one can show that $X(t)$ converges in distribution to a *deterministic* power law, $t^p$ with $p > 0$ when $c < c^*$. Since the $\xi$'s are independent and have uniformly bounded variances for smooth objective functions, the second term converges in distribution to a finite-variance random variable. The first term converges to a finite-variance random variable if $c > c^*$, but to a power of $t$ if $c < c^*$.

## Qualitative Behavior of Schedules

We compare several fixed and adaptive learning rate schedules on a toy stochastic problem. Notice the difficulties that are encountered by some schedules even on a fairly easy problem due to noise in the gradient. The problem is learning a two parameter adaline in the presence of independent uniformly distributed $[-0.5, 0.5]$ noise on the exemplar labels. Exemplars were independently uniformly distributed on $[-1, 1]$. The objective function has a condition number of 10, indicating the presence of the narrow ravine indicated by the elliptical isopleths in the figures. All runs start from the same parameter (weight) vector and receive the same sequence of exemplars. The misadjustment is defined as the Euclidean distance in parameter space to the minimum. Multiples of this quantity bound the usual sum of squares error measure above and below, i.e. sum of squares error is roughly proportional to the misadjustment. Results are presented in figs. 3–8.

## Conclusions

Our empirical tests agree with our theoretical expectations that drift can be used to determine whether the crucial parameter $c$ is large enough. Using this statistic, it will be possible to produce the first fully automatic learning rates which converge at optimal speed. We are currently investigating candidate schedules which we expect to be useful for large-scale LMS, backpropagation, and clustering applications.

**Stochastic Delta-Bar-Delta**

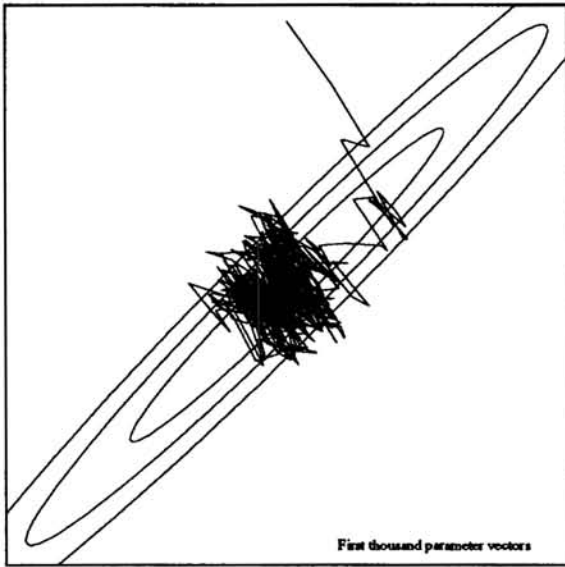
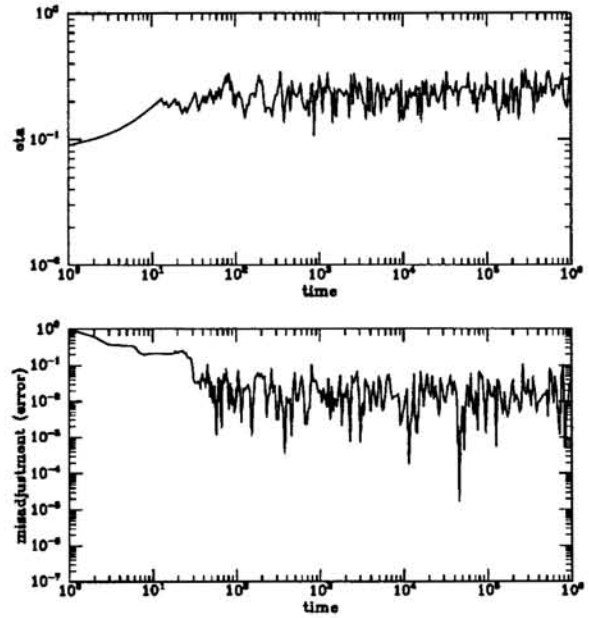

Figure 4: Delta-bar-delta (Jacobs, 1988) was apparently developed for use with deterministic gradient descent. It is also useful for stochastic problems with little noise, which is however not the case for this test problem. In this example $\eta$ *increases* from its initial value, and then stabilizes. We use the algorithm exactly as it appears in Jacobs' paper with noisy gradients substituted for the true gradient (which is unavailable in the stochastic setting). Parameters used were $\eta_0 = 0.1$, $\theta = 0.3$, $\kappa = 0.01$, and $\phi = 0.1$.

**Urasiev**

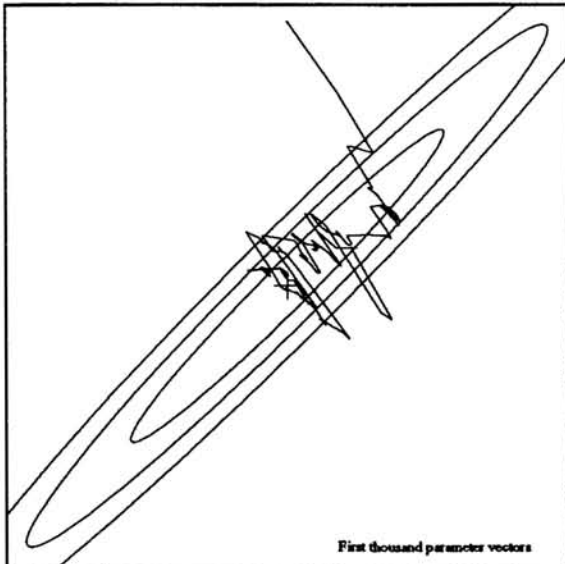
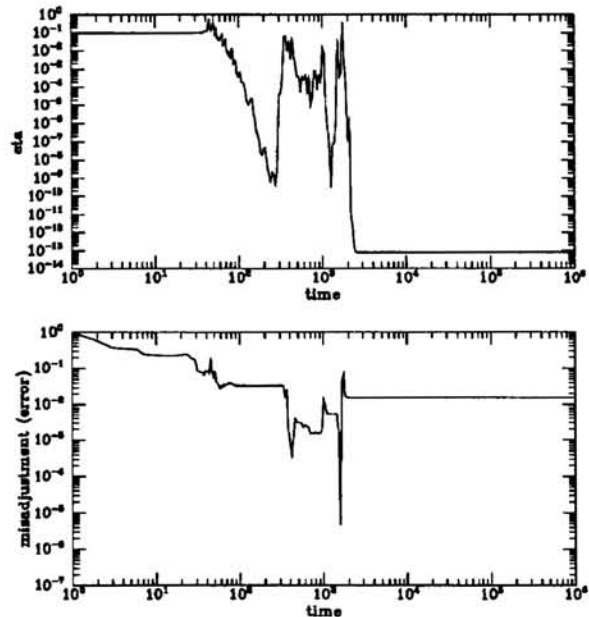

Figure 5: Urasiev's technique (Urasiev, 1988) varies $\eta$ erratically over several orders of magnitude. The large fluctuations apparently cause $\eta$ to completely stop changing after a while due to finite precision effects. Parameters used were $D = 0.2$, $R = 2$, and $U = 1$.

**Fixed Search-Then-Converge, c=c\***

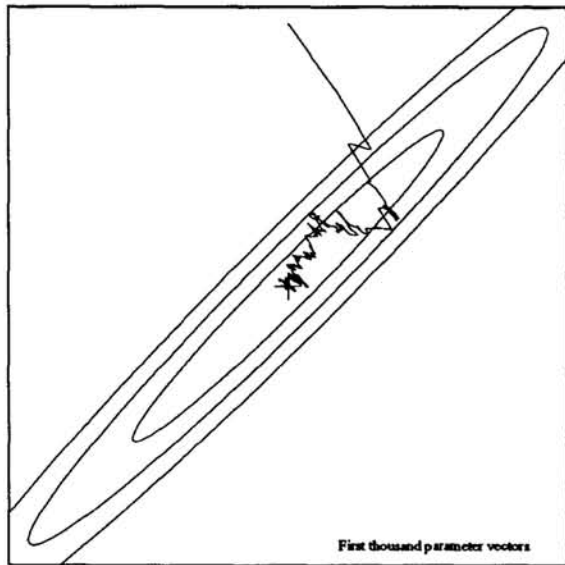
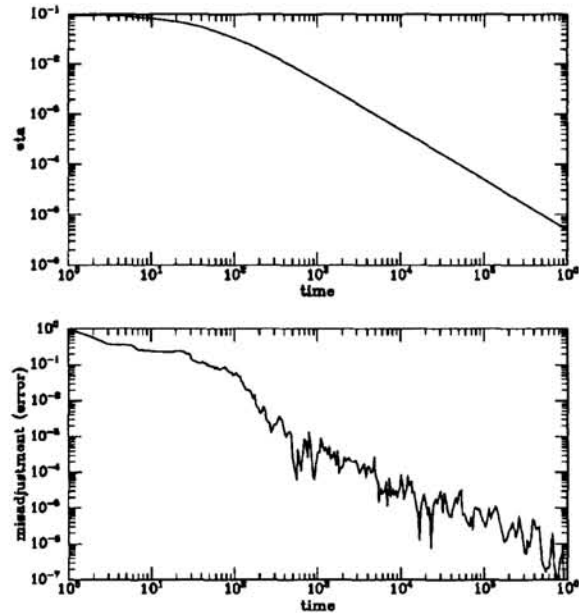

Figure 6: The fixed search-then-converge schedule with $c = c^*$ gives excellent performance. However if $c^*$ is not known, you may get performance as in the next two examples. An adaptive technique is called for.

**Fixed Search-Then-Converge, c=10c\***

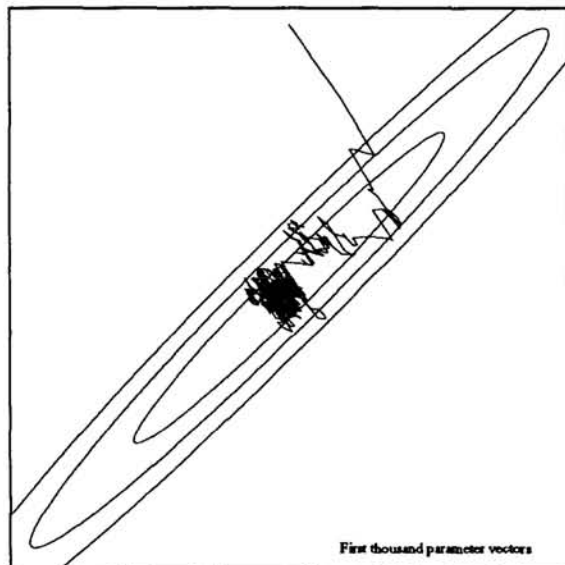
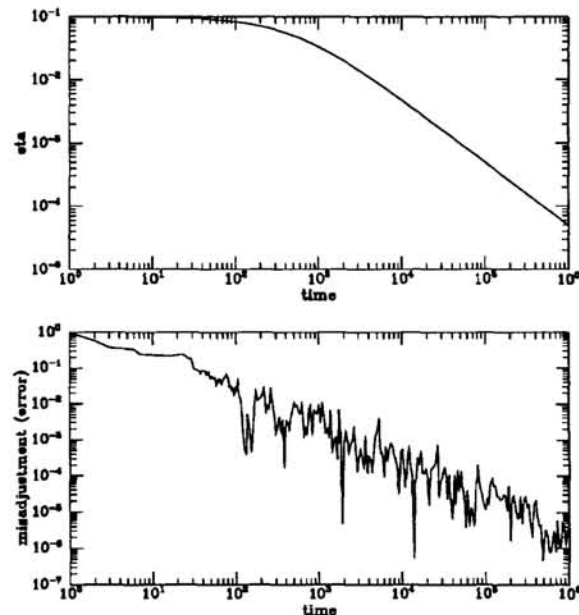

Figure 7: Note that taking $c > c^*$ slows convergence a bit as compared to the $c = c^*$ example in fig. 6, though it could aid escape from bad local minima in a nonlinear problem.

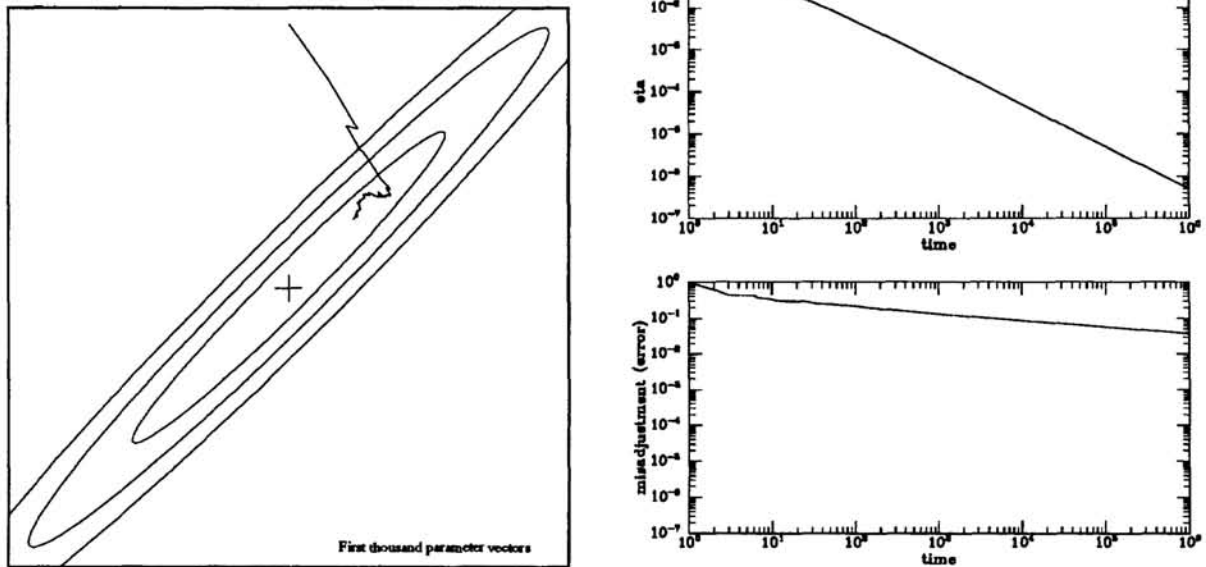

Figure 8: This run illustrates the penalty to be paid if $c < c^*$.

## References

C. Darken and J. Moody. (1990) Note on learning rate schedules for stochastic optimization. *Advances in Neural Information Processing Systems 3*. 832-838.

V.Fabian. (1960) Stochastic approximation methods. *Czechoslovak Math J.* **10** (85) : 123-159.

V. Fabian. (1968) On asymptotic normality in stochastic approximation. *Ann. Math. Stat.* **39**(4):1327-1332.

L. Goldstein. (1987) Mean square optimality in the continuous time Robbins Monro procedure. Technical Report DRB-306. Department of Mathematics, University of Southern California.

R. Jacobs. (1988) Increased rates of convergence through learning rate adaptation. *Neural Networks.* **1**:295-307.

H. Kesten. (1958) Accelerated stochastic approximation. *Annals of Mathematical Statistics.* **29**:41-59.

H. Kushner. (1978) Rates of convergence for sequential Monte Carlo optimization methods. *SIAM J. Control and Optimization.* **16**:150-168.

P. Major and P.Revesz. (1973) A limit theorem for the Robbins-Monro approximation. *Z. Wahrscheinlichkeitstheorie verw. Geb.* **27**:79-86.

S. Urasiev. (1988) Adaptive stochastic quasigradient procedures. In *Numerical Techniques for Stochastic Optimization.* Y. Ermoliev and R. Wets Eds. Springer-Verlag.
